# Incremental and Decremental Support Vector Machine Learning

**Gert Cauwenberghs***
CLSP, ECE Dept.
Johns Hopkins University
Baltimore, MD 21218
*gert@jhu.edu*

**Tomaso Poggio**
CBCL, BCS Dept.
Massachusetts Institute of Technology
Cambridge, MA 02142
*tp@ai.mit.edu*

## Abstract

An on-line recursive algorithm for training support vector machines, one vector at a time, is presented. Adiabatic increments retain the Kuhn-Tucker conditions on all previously seen training data, in a number of steps each computed analytically. The incremental procedure is reversible, and decremental "unlearning" offers an efficient method to exactly evaluate leave-one-out generalization performance. Interpretation of decremental unlearning in feature space sheds light on the relationship between generalization and geometry of the data.

## 1 Introduction

Training a support vector machine (SVM) requires solving a quadratic programming (QP) problem in a number of coefficients equal to the number of training examples. For very large datasets, standard numeric techniques for QP become infeasible. Practical techniques decompose the problem into manageable subproblems over part of the data [7, 5] or, in the limit, perform iterative pairwise [8] or component-wise [3] optimization. A disadvantage of these techniques is that they may give an approximate solution, and may require many passes through the dataset to reach a reasonable level of convergence. An on-line alternative, that formulates the (exact) solution for $\ell+1$ training data in terms of that for $\ell$ data and one new data point, is presented here. The incremental procedure is reversible, and decremental "unlearning" of each training sample produces an exact leave-one-out estimate of generalization performance on the training set.

## 2 Incremental SVM Learning

Training an SVM "incrementally" on new data by discarding all previous data except their support vectors, gives only approximate results [11]. In what follows we consider *incremental learning* as an exact on-line method to construct the solution recursively, one point at a time. The key is to retain the Kuhn-Tucker (KT) conditions on *all* previously seen data, while "adiabatically" adding a new data point to the solution.

### 2.1 Kuhn-Tucker conditions

In SVM classification, the optimal separating function reduces to a linear combination of kernels on the training data, $f(\mathbf{x}) = \sum_j \alpha_j y_j K(\mathbf{x}_j, \mathbf{x}) + b$, with training vectors $\mathbf{x}_i$ and corresponding labels $y_i = \pm 1$. In the dual formulation of the training problem, the

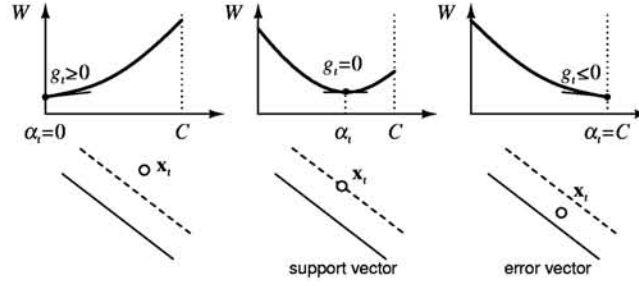

Figure 1: Soft-margin classification SVM training.

coefficients $\alpha_i$ are obtained by minimizing a convex quadratic objective function under constraints [12]

$$\min_{0 \leq \alpha_i \leq C} : \quad W = \frac{1}{2} \sum_{i,j} \alpha_i Q_{ij} \alpha_j - \sum_i \alpha_i + b \sum_i y_i \alpha_i \qquad (1)$$

with Lagrange multiplier (and offset) $b$, and with symmetric positive definite kernel matrix $Q_{ij} = y_i y_j K(\mathbf{x}_i, \mathbf{x}_j)$. The first-order conditions on $W$ reduce to the *Kuhn-Tucker* (KT) conditions:

$$g_i = \frac{\partial W}{\partial \alpha_i} = \sum_j Q_{ij} \alpha_j + y_i b - 1 = y_i f(\mathbf{x}_i) - 1 \quad \begin{cases} \geq 0; & \alpha_i = 0 \\ = 0; & 0 < \alpha_i < C \\ \leq 0; & \alpha_i = C \end{cases} \quad (2)$$

$$\frac{\partial W}{\partial b} = \sum_j y_j \alpha_j = 0 \qquad (3)$$

which partition the training data $D$ and corresponding coefficients $\{\alpha_i, b\}$, $i = 1, \ldots \ell$, in three categories as illustrated in Figure 1 [9]: the set $S$ of *margin support vectors* strictly on the margin ($y_i f(\mathbf{x}_i) = 1$), the set $E$ of *error support vectors* exceeding the margin (not necessarily misclassified), and the remaining set $R$ of (ignored) vectors within the margin.

### 2.2 Adiabatic increments

The margin vector coefficients change value during each incremental step to keep all elements in $D$ in *equilibrium*, *i.e.*, keep their KT conditions satisfied. In particular, the KT conditions are expressed differentially as:

$$\Delta g_i = Q_{ic} \Delta \alpha_c + \sum_{j \in S} Q_{ij} \Delta \alpha_j + y_i \Delta b, \qquad \forall i \in D \cup \{c\} \qquad (4)$$

$$0 = y_c \Delta \alpha_c + \sum_{j \in S} y_j \Delta \alpha_j \qquad (5)$$

where $\alpha_c$ is the coefficient being incremented, initially zero, of a "candidate" vector outside $D$. Since $g_i \equiv 0$ for the margin vector working set $S = \{s_1, \ldots s_{\ell_S}\}$, the changes in coefficients must satisfy

$$\mathcal{Q} \cdot \begin{bmatrix} \Delta b \\ \Delta \alpha_{s_1} \\ \vdots \\ \Delta \alpha_{s_{\ell_S}} \end{bmatrix} = - \begin{bmatrix} y_c \\ Q_{s_1 c} \\ \vdots \\ Q_{s_{\ell_S} c} \end{bmatrix} \Delta \alpha_c \qquad (6)$$

with symmetric but not positive-definite Jacobian $\mathcal{Q}$:

$$\mathcal{Q} = \begin{bmatrix} 0 & y_{s_1} & \cdots & y_{s_{\ell_S}} \\ y_{s_1} & Q_{s_1 s_1} & \cdots & Q_{s_1 s_{\ell_S}} \\ \vdots & \vdots & \ddots & \vdots \\ y_{s_{\ell_S}} & Q_{s_{\ell_S} s_1} & \cdots & Q_{s_{\ell_S} s_{\ell_S}} \end{bmatrix}. \qquad (7)$$

Thus, in equilibrium

$$
\begin{aligned}
\Delta b &= \beta \Delta \alpha_c & (8) \\
\Delta \alpha_j &= \beta_j \Delta \alpha_c, & \forall j \in D & (9)
\end{aligned}
$$

with *coefficient sensitivities* given by

$$
\begin{bmatrix} \beta \\ \beta_{s_1} \\ \vdots \\ \beta_{s_{\ell_S}} \end{bmatrix} = -\mathcal{R} \cdot \begin{bmatrix} y_c \\ Q_{s_1 c} \\ \vdots \\ Q_{s_{\ell_S} c} \end{bmatrix} \tag{10}
$$

where $\mathcal{R} = \mathcal{Q}^{-1}$, and $\beta_j \equiv 0$ for all $j$ outside $S$. Substituted in (4), the margins change according to:

$$
\Delta g_i = \gamma_i \Delta \alpha_c, \qquad \forall i \in D \cup \{c\} \tag{11}
$$

with *margin sensitivities*

$$
\gamma_i = Q_{ic} + \sum_{j \in S} Q_{ij} \beta_j + y_i \beta, \qquad \forall i \notin S \tag{12}
$$

and $\gamma_i \equiv 0$ for all $i$ in $S$.

### 2.3 Bookkeeping: upper limit on increment $\Delta \alpha_c$

It has been tacitly assumed above that $\Delta \alpha_c$ is small enough so that no element of $D$ moves across $S$, $E$ and/or $R$ in the process. Since the $\alpha_j$ and $g_i$ change with $\alpha_c$ through (9) and (11), some bookkeeping is required to check each of the following conditions, and determine the largest possible increment $\Delta \alpha_c$ accordingly:

1. $g_c \leq 0$, with equality when $c$ joins $S$;
2. $\alpha_c \leq C$, with equality when $c$ joins $E$;
3. $0 \leq \alpha_j \leq C, \forall j \in S$, with equality 0 when $j$ transfers from $S$ to $R$, and equality $C$ when $j$ transfers from $S$ to $E$;
4. $g_i \leq 0, \forall i \in E$, with equality when $i$ transfers from $E$ to $S$;
5. $g_i \geq 0, \forall i \in R$, with equality when $i$ transfers from $R$ to $S$.

### 2.4 Recursive magic: $\mathcal{R}$ updates

To add candidate $c$ to the working margin vector set $S$, $\mathcal{R}$ is expanded as:

$$
\mathcal{R} \leftarrow \begin{bmatrix} & & 0 \\ & \mathcal{R} & \vdots \\ & & 0 \\ 0 \cdots 0 & & 0 \end{bmatrix} + \frac{1}{\gamma_c} \begin{bmatrix} \beta \\ \beta_{s_1} \\ \vdots \\ \beta_{s_{\ell_S}} \\ 1 \end{bmatrix} \cdot [\beta, \beta_{s_1} \cdots \beta_{s_{\ell_S}}, 1] \tag{13}
$$

The same formula applies to add *any* vector (not necessarily the candidate) $c$ to $S$, with parameters $\beta$, $\beta_j$ and $\gamma_c$ calculated as (10) and (12).

The expansion of $\mathcal{R}$, as incremental learning itself, is reversible. To remove a margin vector $k$ from $S$, $\mathcal{R}$ is contracted as:

$$
\mathcal{R}_{ij} \leftarrow \mathcal{R}_{ij} - \mathcal{R}_{kk}{}^{-1} \mathcal{R}_{ik} \mathcal{R}_{kj} \qquad \forall i, j \in S \cup \{0\}; i, j \neq k \tag{14}
$$

where index 0 refers to the $b$-term.

The $\mathcal{R}$ update rules (13) and (14) are similar to on-line recursive estimation of the covariance of (sparsified) Gaussian processes [2].

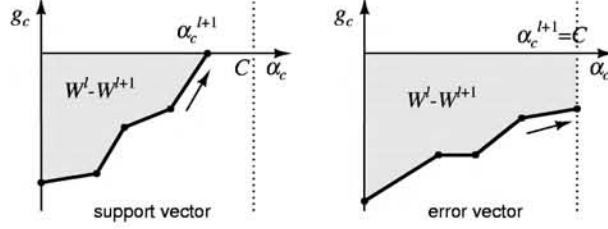

Figure 2: Incremental learning. A new vector, initially for $\alpha_c = 0$ classified with negative margin $g_c < 0$, becomes a new margin or error vector.

## 2.5 Incremental procedure

Let $\ell \to \ell+1$, by adding point $c$ (candidate margin or error vector) to $D$: $D^{\ell+1} = D^\ell \cup \{c\}$. Then the new solution $\{\alpha_i^{\ell+1}, b^{\ell+1}\}$, $i = 1, \ldots \ell + 1$ is expressed in terms of the present solution $\{\alpha_i^\ell, b^\ell\}$, the present Jacobian inverse $\mathcal{R}$, and the candidate $\mathbf{x}_c$, $y_c$, as:

**Algorithm 1 (Incremental Learning, $\ell \to \ell + 1$)**

1. *Initialize $\alpha_c$ to zero;*

2. *If $g_c > 0$, terminate ($c$ is not a margin or error vector);*

3. *If $g_c \leq 0$, apply the largest possible increment $\alpha_c$ so that (the first) one of the following conditions occurs:*

    (a) *$g_c = 0$: Add $c$ to margin set $S$, update $\mathcal{R}$ accordingly, and terminate;*

    (b) *$\alpha_c = C$: Add $c$ to error set $E$, and terminate;*

    (c) *Elements of $D^\ell$ migrate across $S$, $E$, and $R$ ("bookkeeping," section 2.3): Update membership of elements and, if $S$ changes, update $\mathcal{R}$ accordingly.*

    *and repeat as necessary.*

The incremental procedure is illustrated in Figure 2. Old vectors, from previously seen training data, may change status along the way, but the process of adding the training data $c$ to the solution converges in a finite number of steps.

## 2.6 Practical considerations

The trajectory of an example incremental training session is shown in Figure 3. The algorithm yields results identical to those at convergence using other QP approaches [7], with comparable speeds on various datasets ranging up to several thousands training points[1].

A practical on-line variant for larger datasets is obtained by keeping track only of a limited set of "reserve" vectors: $R = \{i \in D \mid 0 < g_i < \epsilon\}$, and discarding all data for which $g_i \geq \epsilon$. For small $\epsilon$, this implies a small overhead in memory over $S$ and $E$. The larger $\epsilon$, the smaller the probability of missing a future margin or error vector in previous data. The resulting storage requirements are dominated by that for the inverse Jacobian $\mathcal{R}$, which scale as $(\ell_S)^2$ where $\ell_S$ is the number of *margin* support vectors, $\#S$.

# 3 Decremental "Unlearning"

Leave-one-out (LOO) is a standard procedure in predicting the generalization power of a trained classifier, both from a theoretical and empirical perspective [12]. It is naturally implemented by *decremental unlearning*, adiabatic reversal of incremental learning, on each of the training data from the full trained solution. Similar (but different) bookkeeping of elements migrating across $S$, $E$ and $R$ applies as in the incremental case.

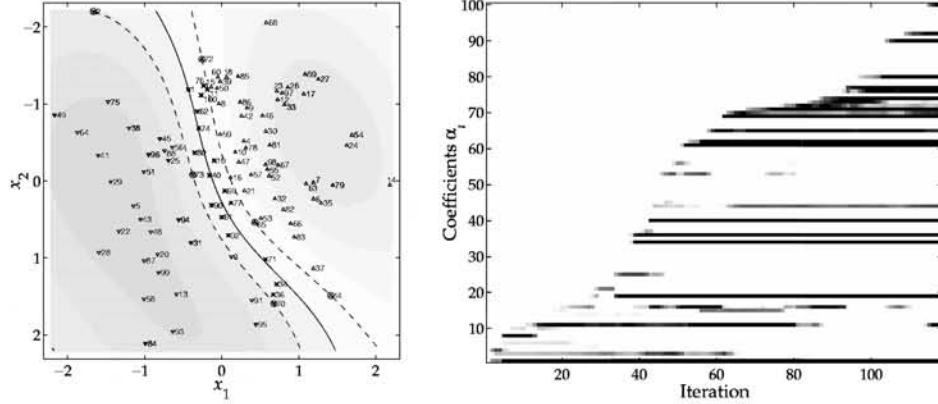

Figure 3: Trajectory of coefficients $\alpha_i$ as a function of iteration step during training, for $\ell = 100$ non-separable points in two dimensions, with $C = 10$, and using a Gaussian kernel with $\sigma = 1$. The data sequence is shown on the left.

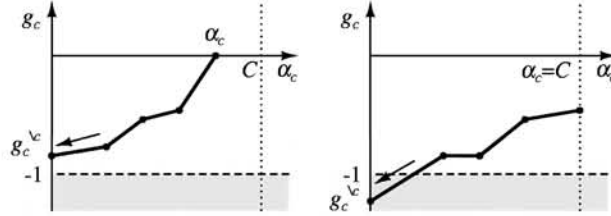

Figure 4: Leave-one-out (LOO) decremental unlearning ($\alpha_c \to 0$) for estimating generalization performance, directly on the training data. $g_c^{\backslash c} < -1$ reveals a LOO classification error.

### 3.1 Leave-one-out procedure

Let $\ell \to \ell - 1$, by removing point $c$ (margin or error vector) from $D$: $D^{\backslash c} = D \setminus \{c\}$. The solution $\{\alpha_i^{\backslash c}, b^{\backslash c}\}$ is expressed in terms of $\{\alpha_i, b\}$, $\mathcal{R}$ and the removed point $\mathbf{x}_c, y_c$. The solution yields $g_c^{\backslash c}$, which determines whether leaving $c$ out of the training set generates a classification error ($g_c^{\backslash c} < -1$). Starting from the full $\ell$-point solution:

**Algorithm 2 (Decremental Unlearning, $\ell \to \ell - 1$, and LOO Classification)**

> 1. *If $c$ is not a margin or error vector: Terminate, "correct" ($c$ is already left out, and correctly classified);*
>
> 2. *If $c$ is a margin or error vector with $g_c < -1$: Terminate, "incorrect" (by default as a training error);*
>
> 3. *If $c$ is a margin or error vector with $g_c \geq -1$, apply the largest possible decrement $\alpha_c$ so that (the first) one of the following conditions occurs:*
>
>     (a) *$g_c < -1$: Terminate, "incorrect";*
>
>     (b) *$\alpha_c = 0$: Terminate, "correct";*
>
>     (c) *Elements of $D^\ell$ migrate across $S$, $E$, and $\mathcal{R}$: Update membership of elements and, if $S$ changes, update $\mathcal{R}$ accordingly.*
>
>     *and repeat as necessary.*

The leave-one-out procedure is illustrated in Figure 4.

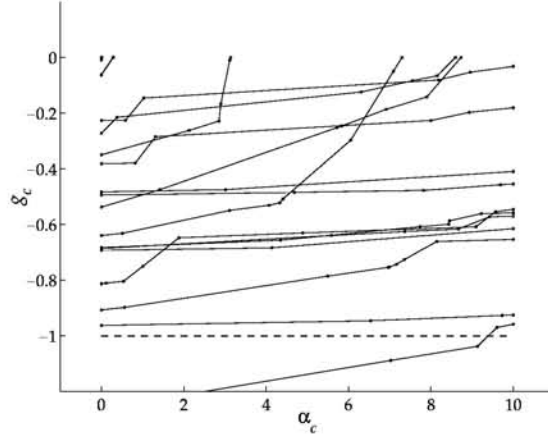

Figure 5: Trajectory of LOO margin $g_c$ as a function of leave-one-out coefficient $\alpha_c$. The data and parameters are as in Figure 3.

### 3.2 Leave-one-out considerations

If an exact LOO estimate is requested, two passes through the data are required. The LOO pass has similar run-time complexity and memory requirements as the incremental learning procedure. This is significantly better than the conventional approach to empirical LOO evaluation which requires $\ell$ (partial but possibly still extensive) training sessions.

There is a clear correspondence between generalization performance and the LOO margin sensitivity $\gamma_c$. As shown in Figure 4, the value of the LOO margin $g_c^{\backslash c}$ is obtained from the sequence of $g_c$ vs. $\alpha_c$ segments for each of the decrement steps, and thus determined by their slopes $\gamma_c$. Incidentally, the LOO approximation using linear response theory in [6] corresponds to the first segment of the LOO procedure, effectively extrapolating the value of $g_c^{\backslash c}$ from the initial value of $\gamma_c$. This simple LOO approximation gives satisfactory results in most (though not all) cases as illustrated in the example LOO session of Figure 5.

Recent work in statistical learning theory has sought improved generalization performance by considering non-uniformity of distributions in feature space [13] or non-uniformity in the kernel matrix eigenspectrum [10]. A geometrical interpretation of decremental unlearning, presented next, sheds further light on the dependence of generalization performance, through $\gamma_c$, on the geometry of the data.

## 4 Geometric Interpretation in Feature Space

The differential Kuhn-Tucker conditions (4) and (5) translate directly in terms of the sensitivities $\gamma_i$ and $\beta_j$ as

$$\gamma_i = Q_{ic} + \sum_{j \in S} Q_{ij}\beta_j + y_i\beta \qquad \forall i \in D \cup \{c\} \tag{15}$$

$$0 = y_c + \sum_{j \in S} y_j\beta_j . \tag{16}$$

Through the nonlinear map $\mathbf{X}_i \equiv y_i\varphi(\mathbf{x}_i)$ into feature space, the kernel matrix elements reduce to linear inner products:

$$Q_{ij} = y_i y_j K(\mathbf{x}_i, \mathbf{x}_j) = \mathbf{X}_i \cdot \mathbf{X}_j, \qquad \forall i, j \tag{17}$$

and the KT sensitivity conditions (15) and (16) in feature space become

$$\gamma_i = \mathbf{X}_i \cdot \left(\mathbf{X}_c + \sum_{j \in S} \mathbf{X}_j\beta_j\right) + y_i\beta \qquad \forall i \in D \cup \{c\} \tag{18}$$

$$0 = y_c + \sum_{j \in S} y_j \beta_j. \tag{19}$$

Since $\gamma_i \equiv 0, \forall i \in S$, (18) and (19) are equivalent to minimizing a functional:

$$\min_{\beta_j} : W_c = \frac{1}{2} (\mathbf{X}_c + \sum_{j \in S} \mathbf{X}_j \beta_j)^2 , \tag{20}$$

subject to the equality constraint (19) with Lagrange parameter $\beta$. Furthermore, the optimal value of $W_c$ immediately yields the sensitivity $\gamma_c$, from (18):

$$\gamma_c = 2W_c = (\mathbf{X}_c + \sum_{j \in S} \mathbf{X}_j \beta_j)^2 \geq 0. \tag{21}$$

In other words, the distance in feature space between sample $c$ and its projection on $S$ along (16) determines, through (21), the extent to which leaving out $c$ affects the classification of $c$. Note that only *margin* support vectors are relevant in (21), and not the error vectors which otherwise contribute to the decision boundary.

## 5 Concluding Remarks

Incremental learning and, in particular, decremental unlearning offer a simple and computationally efficient scheme for on-line SVM training and exact leave-one-out evaluation of the generalization performance on the training data. The procedures can be directly extended to a broader class of kernel learning machines with convex quadratic cost functional under linear constraints, including SV regression. The algorithm is intrinsically on-line and extends to query-based learning methods [1]. Geometric interpretation of decremental unlearning in feature space elucidates a connection, similar to [13], between generalization performance and the distance of the data from the subspace spanned by the margin vectors.

## Footnotes

*On sabbatical leave at CBCL in MIT while this work was performed.

[1] Matlab code and data are available at *http://bach.ece.jhu.edu/pub/gert/svm/incremental*.

## References

[1] C. Campbell, N. Cristianini and A. Smola, "Query Learning with Large Margin Classifiers," in *Proc. 17th Int. Conf. Machine Learning (ICML2000)*, Morgan Kaufman, 2000.

[2] L. Csato and M. Opper, "Sparse Representation for Gaussian Process Models," in *Adv. Neural Information Processing Systems (NIPS'2000)*, vol. **13**, 2001.

[3] T.-T. Frieß, N. Cristianini and C. Campbell, "The Kernel Adatron Algorithm: A Fast and Simple Learning Procedure for Support Vector Machines," in *15th Int. Conf. Machine Learning*, Morgan Kaufman, 1998.

[4] T.S. Jaakkola and D. Haussler, "Probabilistic Kernel Methods," *Proc. 7th Int. Workshop on Artificial Intelligence and Statistics*, 1998.

[5] T. Joachims, "Making Large-Scale Support Vector Machine Learning Practical," in Schölkopf, Burges and Smola, Eds., *Advances in Kernel Methods– Support Vector Learning*, Cambridge MA: MIT Press, 1998, pp 169-184.

[6] M. Opper and O. Winther, "Gaussian Processes and SVM: Mean Field Results and Leave-One-Out," *Adv. Large Margin Classifiers*, A. Smola, P. Bartlett, B. Schölkopf and D. Schuurmans, Eds., Cambridge MA: MIT Press, 2000, pp 43-56.

[7] E. Osuna, R. Freund and F. Girosi, "An Improved Training Algorithm for Support Vector Machines," *Proc. 1997 IEEE Workshop on Neural Networks for Signal Processing*, pp 276-285, 1997.

[8] J.C. Platt, "Fast Training of Support Vector Machines Using Sequential Minimum Optimization," in Schölkopf, Burges and Smola, Eds., *Advances in Kernel Methods– Support Vector Learning*, Cambridge MA: MIT Press, 1998, pp 185-208.

[9] M. Pontil and A. Verri, "Properties of Support Vector Machines," it Neural Computation, vol. **10**, pp 955-974, 1997.

[10] B. Schölkopf, J. Shawe-Taylor, A.J. Smola and R.C. Williamson, "Generalization Bounds via Eigenvalues of the Gram Matrix," *NeuroCOLT*, Technical Report 99-035, 1999.

[11] N.A. Syed, H. Liu and K.K. Sung, "Incremental Learning with Support Vector Machines," in *Proc. Int. Joint Conf. on Artificial Intelligence (IJCAI-99)*, 1999.

[12] V. Vapnik, *The Nature of Statistical Learning Theory*,' New York: Springer-Verlag, 1995.

[13] V. Vapnik and O. Chapelle, "Bounds on Error Expectation for SVM," in Smola, Bartlett, Schölkopf and Schuurmans, Eds., *Advances in Large Margin Classifiers*, Cambridge MA: MIT Press, 2000.
